# A Note on Learning Vector Quantization

**Virginia R. de Sa**
Department of Computer Science
University of Rochester
Rochester, NY 14627

**Dana H. Ballard**
Department of Computer Science
University of Rochester
Rochester, NY 14627

## Abstract

Vector Quantization is useful for data compression. Competitive Learning which minimizes reconstruction error is an appropriate algorithm for vector quantization of unlabelled data. Vector quantization of labelled data for classification has a different objective, to minimize the number of misclassifications, and a different algorithm is appropriate. We show that a variant of Kohonen's LVQ2.1 algorithm can be seen as a multi-class extension of an algorithm which in a restricted 2 class case can be proven to converge to the Bayes optimal classification boundary. We compare the performance of the LVQ2.1 algorithm to that of a modified version having a decreasing window and normalized step size, on a ten class vowel classification problem.

## 1 Introduction

Vector quantization is a form of data compression that represents data vectors by a smaller set of codebook vectors. Each data vector is then represented by its nearest codebook vector. The goal of vector quantization is to represent the data with the fewest codebook vectors while losing as little information as possible.

Vector quantization of unlabelled data seeks to minimize the reconstruction error. This can be accomplished with Competitive learning[Grossberg, 1976; Kohonen, 1982], an iterative learning algorithm for vector quantization that has been shown to perform gradient descent on the following energy function [Kohonen, 1991]

$$\int \|x - w_{s^*(x)}\|^2 p(x)dx.$$

where $p(x)$ is the probability distribution of the input patterns and $w_s$ are the reference or codebook vectors and $s^*(x)$ is defined by $\|x - w_{s^*(x)}\| \leq \|x - w_i\|$ (for all $i$). This minimizes the square reconstruction error of unlabelled data and may work reasonably well for classification tasks if the patterns in the different classes are segregated.

In many classification tasks, however, the different member patterns may not be segregated into separate clusters for each class. In these cases it is more important that members of the same class be represented by the same codebook vector than that the reconstruction error is minimized. To do this, the quantizer can make use of the labelled data to encourage appropriate quantization.

## 2    Previous approaches to Supervised Vector Quantization

The first use of labelled data (or a teaching signal) with Competitive Learning by Rumelhart and Zipser [Rumelhart and Zipser, 1986] can be thought of as assigning a class to each codebook vector and only allowing patterns from the appropriate class to influence each reference vector.

This simple approach is far from optimal though as it fails to take into account interactions between the classes. Kohonen addressed this in his LVQ(1) algorithm[Kohonen, 1986]. He argues that the reference vectors resulting from LVQ(1) tend to approximate for a particular class r,

$$P(x|C_r)P(C_r) - \Sigma_{s \neq r} P(x|C_s)P(C_s).$$

where $P(C_i)$ is the a priori probability of Class i and $P(x|C_i)$ is the conditional density of Class i.

This approach is also not optimal for classification, as it addresses optimal places to put the codebook vectors instead of optimal placement of the *borders* of the vector quantizer which arise from the Voronoi tessellation induced by the codebook vectors. [1]

## 3    Minimizing Misclassifications

In classification tasks the goal is to minimize the numbers of misclassifications of the resultant quantizer. That is we want to minimize:

$$E = \sum_j \int_{D\_R_j} \sum_{i, i \neq j} P(Class_i)P(x|Class_i)dx \qquad (1)$$

where , $P(Class_i)$ is the a priori probability of $Class_i$ and $P(x|Class_i)$ is the conditional density of $Class_i$ and $D\_R_j$ is the decision region for class j (which in this case is all x such that $\|x - w_k\| < \|x - w_i\|$ (for all $i$) and $w_k$ is a codebook vector for class j).

Consider a One-Dimensional problem of two classes and two codebook vectors $w1$ and $w2$ defining a class boundary $b = (w1 + w2)/2$ as shown in Figure 1. In this case Equation 1 reduces to:

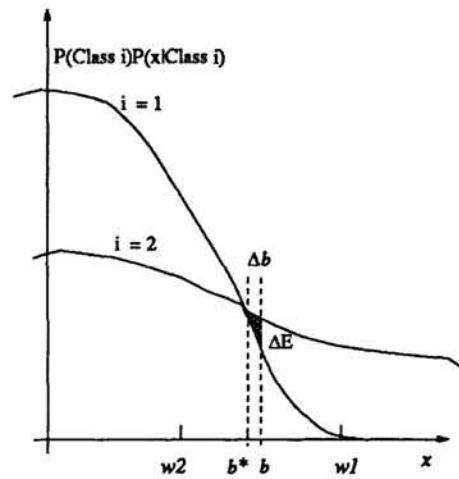

Figure 1: Codebook vectors $w_1$ and $w_2$ define a border $b$. The optimal place for the border is at $b^*$ where $P(C_1)P(x|C_1) = P(C_2)P(x|C_2)$. The extra misclassification errors incurred by placing the border at b is shown by the shaded region.

$$E(b) = \int_{-\infty}^{b} P(C_2)P(x|C_2)dx + \int_{b}^{\infty} P(C_1)P(x|C_1)dx. \tag{2}$$

The derivative of Equation 2 with respect to b is

$$dE/db = P(Class_1)P(b|Class_1) - P(Class_2)P(b|Class_2)$$

That is, the minimum number of misclassifications occurs at $b^*$ where

$$P(Class_1)P(b^*|Class_1) = P(Class_2)P(b^*|Class_2).$$

If $f(x) = (Class_1)P(x|Class_1) - P(Class_2)P(x|Class_2)$ was a regression function then we could use stochastic approximation [Robbins and Monro, 1951] to estimate b* iteratively as

$$b(n + 1) = b(n) + \alpha(n)Z_n$$

where $Z_n$ is a sample of the random variable $Z$ whose expected value is $P(Class_1)P(b(n)|Class_1) - P(Class_2)P(b(n)|Class_2))$ and

$$\lim_{n \to \infty} \alpha(n) = 0$$

$$\Sigma_1^{\infty} \alpha(n) = \infty$$

$$\Sigma_1^{\infty} \alpha^2(n) < \infty$$

However, we do not have immediate access to an appropriate random variable $Z$ but can express $P(Class_1)P(x|Class_1) - P(Class_2)P(x|Class_2)$ as the limit of a sequence of regression functions using the Parzen Window technique. In the Parzen window technique, probability density functions are estimated as the sum of appropriately normalized pulses centered at

the observed values. More formally, we can estimate $P(x|Class_i)$ as [Sklansky and Wassel, 1981]

$$\hat{P}_i^n(x) = \frac{1}{n} \sum_{j=1}^{n} \Psi_n(x - X_j, c_n)$$

where $X_j$ is the sample data point at time j, and $\Psi_n(x - z, c(n))$ is a Parzen window function centred at $z$ with width parameter $c(n)$ that satisfies the following conditions

$$\Psi_n(x - z, c(n)) \geq 0, \forall x, z$$

$$\int_{-\infty}^{\infty} \Psi_n(x - z, c(n)) dx = 1$$

$$\lim_{n \to \infty} \frac{1}{n} \int_{-\infty}^{\infty} \Psi_n^2(x - z, c(n)) dx = 0$$

$$\lim_{n \to \infty} \Psi_n(x - z, c(n)) = \delta(x - z)$$

We can estimate $f(x) = P(Class_1)P(x|Class_1) - P(Class_2)P(x|Class_2)$ as

$$\hat{f}^n(x) = \frac{1}{n} \sum_{j=1}^{n} S(X_j)\Psi_n(x - X_j, c(n))$$

where $S(X_j)$ is $+1$ if $X_j$ is from $Class_1$ and $-1$ if $X_j$ is from $Class_2$.

Then

$$\lim_{n \to \infty} \hat{f}^n(x) = P(Class_1)P(x|Class_1) - P(Class_2)P(x|Class_2)$$

and

$$\lim_{n \to \infty} E[S(X)\Psi_n(x - X, c(n)] = P(Class_1)P(x|Class_1) - P(Class_2)P(x|Class_2)$$

Wassel and Sklansky [1972] have extended the stochastic approximation method of Robbins and Monro [1951] to find the zero of a function that is the limit of a sequence of regression functions and show rigourously that for the above case (where the distribution of $Class_1$ is to the left of that of $Class_2$ and there is only one crossing point) the stochastic approximation procedure

$$b(n + 1) = b(n) + \alpha(n)Z_n(x_n, Class(n), b(n), c(n)) \tag{3}$$

using

$$Z_n = \begin{cases} 2c(n)\Psi(X_n - b(n), c(n)) & \text{for } X_n \in Class1 \\ -2c(n)\Psi(X_n - b(n), c(n)) & \text{for } X_n \in Class2 \end{cases}$$

converges to the Bayes optimal border with probability one where $\Psi(x - b, c)$ is a Parzen window function. The following standard conditions for stochastic approximation convergence are needed in their proof

$$\alpha(n), c(n) > 0, \qquad \lim_{n \to \infty} c(n) = 0 \qquad \lim_{n \to \infty} \alpha(n) = 0,$$

$$\Sigma_1^\infty \alpha(n)c(n) = \infty, \qquad \Sigma_1^\infty \alpha(n)^2 c(n)^2 < \infty$$

as well as a condition that for rectangular Parzen functions reduces to a requirement that $P(Class_1)P(x|Class_1) - P(Class_2)P(x|Class_2)$ be strictly positive to the left of $b^*$ and strictly negative to the right of $b^*$ (for full details of the proof and conditions see [Wassel and Sklansky, 1972]).

The above argument has only addressed the motion of the border. But $b$ is defined as $b = (w1 + w2)/2$, thus we can move the codebook vectors according to

$$dE/dw1 = dE/dw2 = .5dE/db.$$

We could now write Equation 3 as

$$w_i(n+1) = w_i(n) + \alpha_2(n)\frac{(X_n - w_i(n-1))}{|X_n - w_i(n-1)|}$$

$$w_j(n+1) = w_j(n) - \alpha_2(n)\frac{(X_n - w_j(n-1))}{|X_n - w_j(n-1)|}$$

if $X_n$ lies in window of width $2c(n)$ centred at $b(n)$, otherwise

$$w_i(n+1) = w_i(n), \qquad w_j(n+1) = w_j(n)$$

where we have used rectangular Parzen window functions and $X_n$ is from $Class_i$. This holds if $Class_1$ is to the right or left of $Class_2$ as long as $w_1$ and $w_2$ are relatively ordered appropriately.

Expanding the problem to more dimensions, and more classes with more codebook vectors per class, complicates the analysis as a change in two codebook vectors to better adjust their border affects more than just the border between the two codebook vectors. However ignoring these effects for a first order approximation suggests the following update procedure:

$$w_i^*(n) = w_i^*(n-1) + \alpha(n)\frac{(X_n - w_i^*(n-1))}{\|X_n - w_i^*(n-1)\|}$$

$$w_j^*(n) = w_j^*(n-1) - \alpha(n)\frac{(X_n - w_j^*(n-1))}{\|X_n - w_j^*(n-1)\|}$$

where $\alpha(n)$ obeys the constraints above, $X_n$ is from $Class_i$, and $w_i^*, w_j^*$ are the two nearest codebook vectors, one each from class i and j ($j \neq i$) and $x_n$ lies within $c(n)$ of the border between them. (No changes are made if all the above conditions are not true). As above this algorithm assumes that the initial positions of the codebook vectors are such that they will not have to cross during the algorithm.

The above algorithm is similar to Kohonen's LVQ2.1 algorithm (which is performed after appropriate initialization of the codebook vectors) except for the normalization of the step size, the decreasing size of the window width c(n) and constraints on the learning rate $\alpha$.

## 4 Simulations

Motivated by the theory above, we decided to modify Kohonen's LVQ2.1 algorithm to add normalization of the step size and a decreasing window. In order to allow closer comparison with LVQ2.1, all other parts of the algorithm were kept the same. Thus $\alpha$ decreased linearly. We used a linear decrease on the window size and defined it as in LVQ2.1 for easier parameter matching. For a window size of $w$ all input vectors satisfying $d_i/d_j > \frac{(1-w)}{(1+w)}$ where $d_i$ is the distance to the closest codebook vector and $d_j$ is the distance to the next closest codebook vector, fall into the window between those two vectors (Note however, that updates only occur if the two closest codebook vectors belong to different classes).

The data used is a version of the Peterson and Barney vowel formant data [2]. The dataset consists of the first and second formants for ten vowels in a /hVd/ context from 75 speakers (32 males, 28 females, 15 children) who repeated each vowel twice [3]. As we were not testing generalization , the training set was used as the test set.

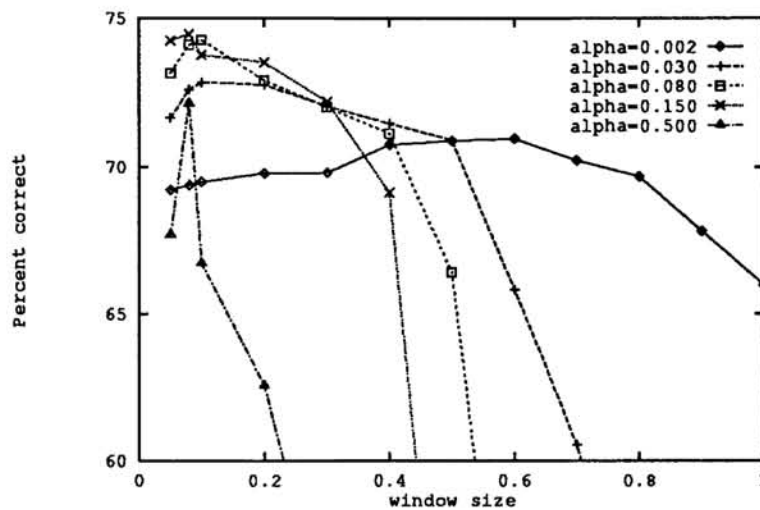

Figure 2: The effect of different window sizes on the accuracy for different values of initial $\alpha$.

We ran three sets of experiments varying the number of codebook vectors and the number of pattern presentations. For the first set of experiments there were 20 codebook vectors and the algorithms ran for 40000 steps. Figure 2 shows the effect of varying the window size for different initial learning rates $\alpha(1)$ in the LVQ2.1 algorithm. The values plotted are averaged over three runs (The order of presentation of patterns is different for the different runs). The sensitivity of the algorithm to the window size as mentioned in [Kohonen, 1990] is evident. In general we found that as the learning rate is increased the peak accuracy is improved at the expense of the accuracy for other window widths. After a certain value

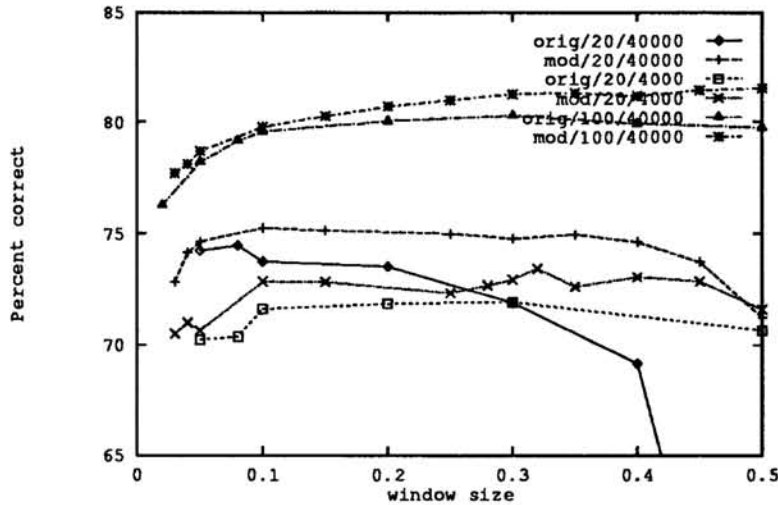

Figure 3: The performance of LVQ2.1 with and without the modifications (normalized step size and decreasing window) for 3 different conditions. The legend gives in order [the alg type/ the number of codebook vectors/ the number of pattern presentations]

the accuracy declines for further increases in learning rate.

Figure 3 shows the improvement achieved with normalization and a linearly decreasing window size for three sets of experiments : (20 code book vectors/40000 pattern presentations), (20 code book vectors/ 4000 pattern presentations) and (100 code book vectors/40000 pattern presentations). For the *decreasing window* algorithm, the x-axis represents the window size in the middle of the run. As above, the values plotted were averaged over three runs. The values of $\alpha(1)$ were the same within each algorithm over all three conditions. A graph using the best $\alpha$ found for each condition separately is almost identical. The graph shows that the modifications provide a modest but consistent improvement in accuracy across the conditions.

In summary the preliminary experiments indicate that a decreasing window and normalized step size can be worthwhile additions to the LVQ2.1 algorithm and further experiments on the generalization properties of the algorithm and with other data sets may be warranted. For these tests we used a linear decrease of the window size and learning rate to allow for easier comparison with the LVQ2.1 algorithm. Further modifications on the algorithm that experiment with different functions (that obey the theoretical constraints) for the learning rate and window size decrease may result in even better performance.

## 5   Summary

We have shown that Kohonen's LVQ2.1 algorithm can be considered as a variant on a generalization of an algorithm which is optimal for a 1Dimensional/2 codebook vector problem. We added a decreasing window and normalized step size, suggested from the one dimensional algorithm, to the LVQ2.1 algorithm and found a small but consistent improvement in accuracy.

**Acknowledgements**

We would like to thank Steven Nowlan for his many helpful suggestions on an earlier draft and for making the vowel formant data available to us. We are also grateful to Leonidas Kontothanassis for his help in coding and discussion. This work was supported by a grant from the Human Frontier Science Program and a Canadian NSERC 1967 Science and Engineering Scholarship to the first author who also received A NIPS travel grant to attend the conference.

## Footnotes

[1] Kohonen [1986] showed this by showing that the use of a "weighted" Voronoi tessellation (where the relative distances of the borders from the reference vectors was changed) worked better. However no principled way to calculate the relative weights was given and the application to real data used the unweighted tessellation.

[2]obtained from Steven Nowlan

[3]3 speakers were missing one vowel and the raw data was linearly transformed to have zero mean and fall within the range [−3, 3] in both components

# References

[Grossberg, 1976] Stephen Grossberg, "Adaptive Pattern Classification and Universal Recoding: I. Parallel Development and Coding of Neural Feature Detectors," *Biological Cybernetics*, 23:121–134, 1976.

[Kohonen, 1982] Teuvo Kohonen, "Self-Organized Formation of Topologically Correct Feature Maps," *Biological Cybernetics*, 43:59–69, 1982.

[Kohonen, 1986] Teuvo Kohonen, "Learning Vector Quantization for Pattern Recognition," Technical Report TKK-F-A601, Helsinki University of Technology, Department of Technical Physics, Laboratory of Computer and Information Science, November 1986.

[Kohonen, 1990] Teuvo Kohonen, "Statistical Pattern Recognition Revisited," In R. Eckmiller, editor, *Advanced Neural Computers*, pages 137–144. Elsevier Science Publishers, 1990.

[Kohonen, 1991] Teuvo Kohonen, "Self-Organizing Maps: Optimization Approaches," In T. Kohonen, K. Makisära, O. Simula, and J. Kangas, editors, *Artificial Neural Networks*, pages 981–990. Elsevier Science Publishers, 1991.

[Robbins and Monro, 1951] Herbert Robbins and Sutton Monro, "A Stochastic Approximation Method," *Annals of Math. Stat.*, 22:400—407, 1951.

[Rumelhart and Zipser, 1986] D. E. Rumelhart and D. Zipser, "Feature Discovery by Competitive Learning," In David E. Rumelhart, James L. McClelland, and the PDP Research Group, editors, *Parallel Distributed Processing: Explorations in the Microstructure of Cognition*, volume 2, pages 151–193. MIT Press, 1986.

[Sklansky and Wassel, 1981] Jack Sklansky and Gustav N. Wassel, *Pattern Classifiers and Trainable Machines*, Springer-Verlag, 1981.

[Wassel and Sklansky, 1972] Gustav N. Wassel and Jack Sklansky, "Training a One-Dimensional Classifier to Minimize the Probability of Error," *IEEE Transactions on Systems, Man, and Cybernetics*, SMC-2(4):533—541, 1972.